# Model Based Population Tracking and Automatic Detection of Distribution Changes

**Igor V. Cadez** *
Dept. of Information and Computer Science,
University of California,
Irvine, CA 92612
*icadez@ics.uci.edu*

**P. S. Bradley**
digiMine, Inc.
10500 NE 8th Street,
Bellevue, WA 98004-4332
*paulb@digimine.com*

## Abstract

Probabilistic mixture models are used for a broad range of data analysis tasks such as clustering, classification, predictive modeling, etc. Due to their inherent probabilistic nature, mixture models can easily be combined with other probabilistic or non-probabilistic techniques thus forming more complex data analysis systems. In the case of online data (where there is a stream of data available) models can be constantly updated to reflect the most current distribution of the incoming data. However, in many business applications the models themselves represent a parsimonious summary of the data and therefore it is not desirable to change models frequently, much less with every new data point. In such a framework it becomes crucial to track the applicability of the mixture model and detect the point in time when the model fails to adequately represent the data. In this paper we formulate the problem of change detection and propose a principled solution. Empirical results over both synthetic and real-life data sets are presented.

## 1  Introduction and Notation

Consider a data set $D = \{x_1, x_2, \ldots, x_n\}$ consisting of $n$ independent, identically distributed (iid) data points. In context of this paper the data points could be vectors, sequences, etc. Further, consider a probabilistic mixture model that maps each data set to a real number, the probability of observing the data set:

$$P(D|\Theta) = \prod_{i=1}^{n} P(x_i|\Theta) = \prod_{i=1}^{n} \sum_{k=1}^{K} \pi_k P(x_i|\theta_k), \qquad (1)$$

where the model is parameterized by $\Theta = \{\pi_1, \ldots, \pi_K, \theta_1, \ldots, \theta_K\}$. Each $P(.|\theta_k)$ represents a *mixture component*, while $\pi_i$ represents *mixture weights*. It is often more convenient

to operate with the log of the probability and define the *log-likelihood* function as:

$$l(\Theta|D) = \log P(D|\Theta) = \sum_{i=1}^{n} \log P(x_i|\Theta) = \sum_{i=1}^{n} Log P_i$$

which is *additive* over data points rather than multiplicative. The $Log P_i$ terms we introduce in the notation represent each data point's contribution to the overall log-likelihood and therefore describe how well a data point fits under the model. For example, Figure 3 shows a distribution of $Log P$ scores using a mixture of conditionally independent (CI) models.

Maximizing probability[1] of the data with respect to the parameters $\Theta$ can be accomplished by the *Expectation-Maximization (EM)* algorithm [6] in linear time in both data complexity (e.g., number of dimensions) and data set size (e.g., number of data points). Although EM guarantees only local optimality, it is a preferred method for finding good solutions in linear time. We consider an arbitrary but fixed parametric form of the model, therefore we sometimes refer to a specific set of parameters $\Theta$ as the *model*. Note that since the logarithm is a monotonic function, the optimal set of parameters is the same whether we use likelihood or log-likelihood.

Consider an online data source where there are data sets $D_t$ available at certain time intervals $t$ (not necessarily equal time periods or number of data points). For example, there could be a data set generated on a daily basis, or it could represent a constant stream of data from a monitoring device. In addition, we assume that we have an initial model $\Theta_0$ that was built (optimized, fitted) on some in-sample data $D^0 = \{D_1, D_2, \ldots, D_{t_0}\}$. We would like to be able to detect a change in the underlying distribution of data points within data sets $D_t$ that would be sufficient to require building of a new model $\Theta_1$. The criterion for building a new model is loosely defined as "the model does not adequately fit the data anymore".

## 2 Model Based Population Similarity

In this section we formulate the problem of model-based population similarity and tracking. In case of mixture models we start with the following observations:

- The mixture model defines the probability density function (PDF) that is used to score each data point ($Log P$ scores), leading to the score for the overall population (log-likelihood or sum of $Log P$ scores).

- The optimal mixture model puts more PDF mass over dense regions in the data space. Different components allow the mixture model to distribute its PDF over disconnected dense regions in the data space. More PDF mass in a portion of the data space implies higher $Log P$ scores for the data points lying in that region of the space.

- If model is to generalize well (e.g., there is no significant overfitting) it cannot put significant PDF mass over regions of data space that are populated by data points solely due to the details of a specific data sample used to build the model.

- Dense regions in the data space discovered by a non-overfitting model are the intrinsic property of the true data-generating distribution even if the functional form of the model is not well matched with the true data generating distribution. In the latter case, the model might not be able to discover *all* dense regions or might not model the correct *shape* of the regions, but the regions that are discovered (if any) are intrinsic to the data.

- If there is confidence that the model is not overfitting and that it generalizes well (e.g., cross-validation was used to determine the optimal number of mixture components), the new data from the same distribution as the in-sample data should be dense in the same regions that are *predicted* by the model.

Given these observations, we seek to define a measure of data-distribution similarity based on how well the dense regions of the data space are preserved when new data is introduced. In model based clustering, *dense* regions are equivalent to *higher $LogP$* scores, hence we cast the problem of determining data distribution similarity into one of determining $LogP$ distribution similarity (relative to the model). For example, Figure 3 (left) shows a histogram of one such distribution. It is important to note several properties of Figure 3: 1) there are several distinct peaks from which distribution tails off toward smaller $LogP$ values, therefore simple summary scores fail to efficiently summarize the $LogP$ distribution. For example, log-likelihood is proportional to the mean of $LogP$ distribution in Figure 3, and the mean is not a very useful statistic when describing such a multimodal distribution (also confirmed experimentally); 2) the histogram itself is not a truly non-parametric representation of the underlying distribution, given that the results are dependent on bin width. In passing we also note that the shape of the histogram in Figure 3 is a consequence of the CI model we use: different peaks come from different discrete attributes, while the tails come from continuous Gaussians. It is a simple exercise to show that $LogP$ scores for a 1-dimensional data set generated by a single Gaussian have an exponential distribution with a sharp cutoff on the right and tail toward the left.

To define the similarity of the data distributions based on $LogP$ scores in a purely non-parametric way we have at our disposal the powerful formalism of Kolmogorov-Smirnov (KS) statistics [7]. KS statistics make use of empirical cumulative distribution functions (CDF) to estimate distance between two empirical 1-dimensional distributions, in our case distributions of $LogP$ scores. In principle, we could compare the $LogP$ distribution of the new data set $D_t$ to that of the training set $D^0$ and obtain the probability that the two came from the same distribution. In practice, however, this approach is not feasible since we do not assume that the estimated model and the true data generating process share the same functional form (see Section 3). Consequently, we need to consider the specific KS score in relation to the *natural variability* of the true data generating distribution. In the situation with streaming data, the model is estimated over the in-sample data $D^0$. Then the individual in-sample data sets $D_1, D_2, \ldots, D_{t_0}$ are used to estimate the natural variability of the KS statistics. This variability needs to be quantified due to the fact that the model may not truly match the data distribution. When the natural variance of the KS statistics over the in-sample data has been determined, the $LogP$ scores for a new dataset $D_t$, $t > t_0$ are computed. Using principled heuristics, one can then determine whether or not the $LogP$ *signature* for $D_t$ is significantly different than the $LogP$ signatures for the in-sample data.

To clarify various steps, we provide an algorithmic description of the change detection process.

**Algorithm 1 (Quantifying Natural Variance of KS Statistics):**
Given an "in-sample" dataset $D^0 = \{D_1, D_2, \ldots, D_{t_0}\}$, proceed as follows:

1. Estimate the parameters $\Theta_0$ of the mixture model $P(D|\Theta)$ over $D^0$ (see equation (1)).

2. Compute

$$LogP(D_i) = \sum_{\hat{i}=1}^{n_i} \log P(x_{\hat{i}}|\Theta_0), \ x_{\hat{i}} \in D_i, n_i = |D_i|, i = 1, \ldots, t_0. \quad (2)$$

3. For $1 \leq i, j \leq t_0$, compute $L_{\text{KS}}(i, j) = \log [P_{\text{KS}}(D_i, D_j)]$. See [7] for details on $P_{\text{KS}}$ computation.

4. For $1 \leq i \leq t_0$, compute the KS measure $M_{\text{KS}}(i)$ as

$$M_{\text{KS}}(i) = \frac{\sum_{j=1}^{t_0} L_{\text{KS}}(i,j)}{t_0}. \tag{3}$$

5. Compute $\mu_M = Mean[M_{\text{KS}}(i)]$ and $\sigma_M = STD[M_{\text{KS}}(i)]$ to quantify the natural variability of $M_{\text{KS}}$ over the "in-sample" data.

**Algorithm 2 (Evaluating New Data):**
Given a new dataset $D_t$, $t > t_0$, $\mu_M$ and $\sigma_M$ proceed as follows:

1. Compute $LogP(D_t)$ as in (2).
2. For $1 \leq i \leq t_0$, compute $L_{\text{KS}}(i,t)$.
3. Compute $M_{\text{KS}}(t)$ as in (3).
4. Apply decision criteria using $M_{\text{KS}}(t)$, $\mu_M$, $\sigma_M$ to determine whether or not $\Theta_0$ is a good fit for the new data. For example, if

$$\frac{|M_{\text{KS}}(t) - \mu_M|}{\sigma_M} > 3, \tag{4}$$

then $\Theta_0$ is not a good fit any more.

Note, however, that the 3-$\sigma$ interval be interpreted as a confidence interval only in the limit when number of data sets goes to infinity. In applications presented in this paper we certainly do not have that condition satisfied and we consider this approach as an "educated heuristic" (gaining firm statistical grounds in the limit).

## 2.1 Space and Time Complexity of the Methodology

The proposed methodology was motivated by a business application with large data sets, hence it must have time complexity that is close to linear in order to scale well. In order to assess the time complexity, we use the following notation: $n_t = |D_t|$ is the number of data points in the data set $D_t$; $t_0$ is the index of the last in-sample data set, but is also the number of in-sample data sets; $n_0 = |D^0| = \sum_{t=1}^{t_0} n_t$ is the total number of in-sample data points (in all the in-sample data sets); $\overline{n} = n_0/t_0$ is the average number of data points in the in-sample data sets. For simplicity of argument, we assume that all the data sets are approximately of the same size, that is $n_t \approx \overline{n}$.

The analysis presented here does not take into account the time and space complexity needed to estimated the parameters $\Theta$ of the mixture model (1). In the first phase of the methodology, we must score each of the in-sample data points under the model (to obtain the $LogP$ distributions) which has time complexity of $O(n_0)$. Calculation of KS statistics for two data sets is done in one pass over the $LogP$ distributions, but it requires that the $LogP$ scores be sorted, hence it has time complexity of $2\overline{n} + 2O(\overline{n} \log \overline{n}) = O(\overline{n} \log \overline{n})$. Since we must calculate all the pairwise KS measures, this step has time complexity of $t_0(t_0 - 1)/2 \, O(\overline{n} \log \overline{n}) = O(t_0^2 \overline{n} \log \overline{n})$. In-sample mean and variance of the KS measure are obtained in time which is linear in $t_0$ hence the asymptotic time complexity does not change. In order to evaluate out-of-sample data sets we must keep $LogP$ distributions for each of the in-sample data sets as well as several scalars (e.g., mean and variance of the in-sample KS measure) which requires $O(n_0)$ memory.

To score an out-of-sample data set $D_t$, $t > t_0$, we must first obtain the $LogP$ distribution of $D_t$ which has time complexity of $O(\overline{n})$ and then calculate the KS measure relative to each of the in-sample data sets which has time complexity $O(\overline{n} \log \overline{n})$ per in-sample data set, or $t_0 O(\overline{n} \log \overline{n}) = O(t_0 \overline{n} \log \overline{n})$ for the full in-sample period. The $LogP$ distribution for $D_t$ can be discarded once the pairwise KS measures are obtained.

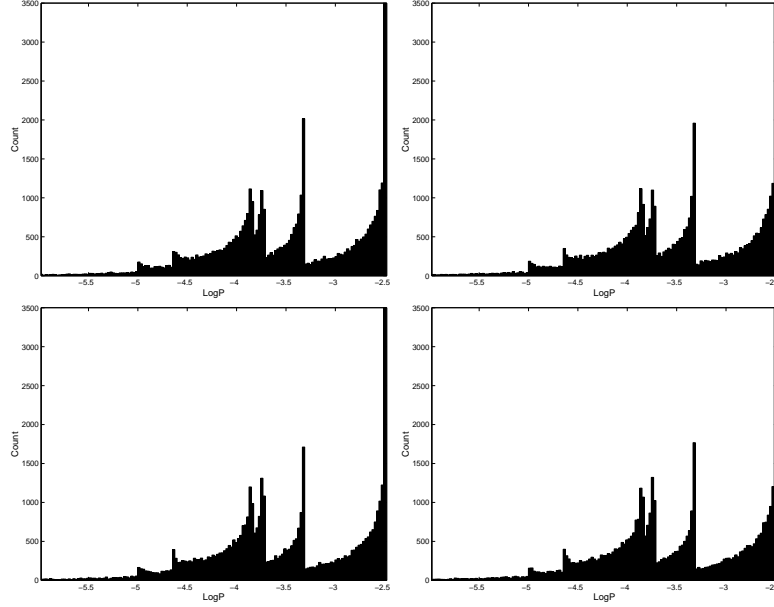

Figure 1: Histograms of $LogP$ scores for two data sets generated from the first model (top row) and two data sets generated from the second model (bottom row). Each data set contains 50,000 data points. All histograms are obtained from the model fitted on the in-sample period.

Overall, the proposed methodology requires $O(n_0)$ memory, $O(t_0^2 \overline{n} \log \overline{n})$ time for prepro-cessing and $O(t_0 \overline{n} \log \overline{n})$ time for out-of-sample evaluation. Further, since $t_0$ is typically a small constant (e.g., $t_0 = 7$ or $t_0 = 30$), the out-of-sample evaluation practically has time complexity of $O(\overline{n} \log \overline{n})$.

# 3 Experimental Setup

Experiments presented consist of two parts: experiments on synthetic data and experiments on the aggregations over real web-log data.

## 3.1 Experiments on Synthetic Data

Synthetic data is a valuable tool when determining both applicability and limitations of the proposed approach. Synthetic data was generated by sampling from a a two component CI model (the true model is not used in evaluations). The data consist of a two-state discrete dimension and a continuous dimension. First 100 data sets where generated by sampling from a mixture model with parameters: $[\pi_1, \pi_2] = [0.6, 0.4]$ as weights, $\underline{\theta}_1 = [0.8, 0.2]$ and $\underline{\theta}_2 = [0.4, 0.6]$ as discrete state probabilities, $[\mu_1, \sigma_1^2] = [10, 5]$ and $[\mu_2, \sigma_2^2] = [0, 7]$ as mean and variance (Gaussian) for the continuous variable. Then the discrete dimension probability of the second cluster was changed from $\underline{\theta}_2 = [0.4, 0.6]$ to $\underline{\theta}'_2 = [0.5, 0.5]$ keeping the remaining parameters fixed and an additional 100 data sets were generated by sampling from this altered model. This is a fairly small change in the distribution and the underlying $LogP$ scores appear to be very similar as can be seen in Figure 1. The figure shows $LogP$ distributions for the first two data sets generated from the first model (top row) and the first two data sets generated from the second model (bottom row). Plots within each

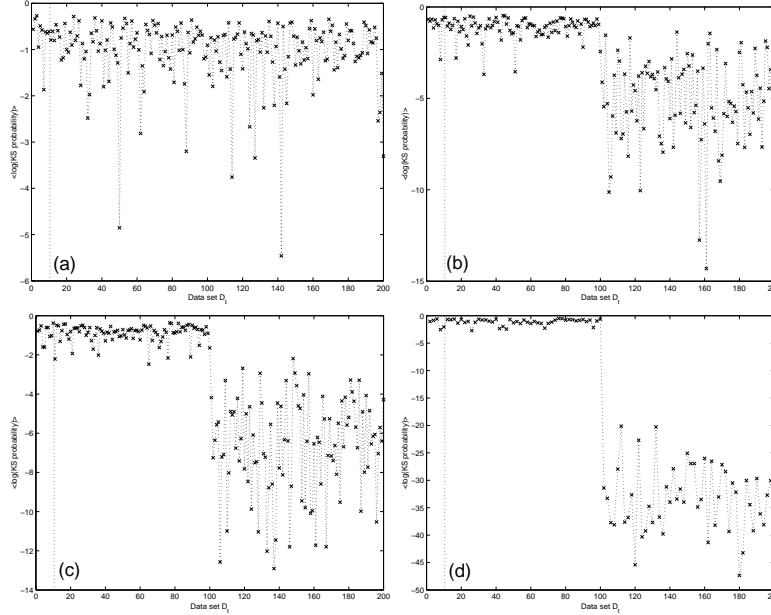

Figure 2: Average log(KS probability) over the in-sample period for four experiments on synthetic data, varying the number of data points per data set: a) 1,000; b) 5,000; c) 10,000; d) 50,000. The dotted vertical line separates in-sample and out-of-sample periods. Note that $y$-axes have different scales in order to show full variability of the data.

row should be more similar than plots from different rows, but this is difficult to discern by visual inspection.

Algorithms 1 and 2 were evaluated by using the first 10 data sets to estimate a two component model. Then pairwise KS measures were calculated between all possible data set pairs relative to the estimated model. Figure 2 shows average KS measures over in-sample data sets (first 10) for four experiments varying the number of data points in each experiment. Note that the vertical axes are different in each of the plots to better show the range of values. As the number of data points in the data set increases, the change that occurs at $t = 101$ becomes more apparent. At 50,000 data points (bottom right plot of Figure 2) the change in the distribution becomes easily detectable. Since this number of data points is typically considered to be small compared to the number of data points in our real life applications we expect to be able to detect such slight distribution changes.

## 3.2   Experiments on Real Life Data

Figure 3 shows a distribution for a typical day from a content web-site. There are almost 50,000 data points in the data set with over 100 dimensions each. The $LogP$ score distribution is similar to that of synthetic data in Figure 1 which is a consequence of the CI model used. Note, however, that in this data set the true generating distribution is not known and is unlikely to be purely a CI model. Therefore, the average log KS measure over in-sample data has much lower values (see Figure 3 right, and plots in Figure 2). Another way to phrase this observation is to note that since the true generating data distribution is most likely not CI, the observed similarity of $LogP$ distributions (the KS measure) is much lower since there are two factors of dissimilarity: 1) different data sets; 2) inability of the CI model to capture all the aspects of the true data distribution. Nonetheless, the first 31

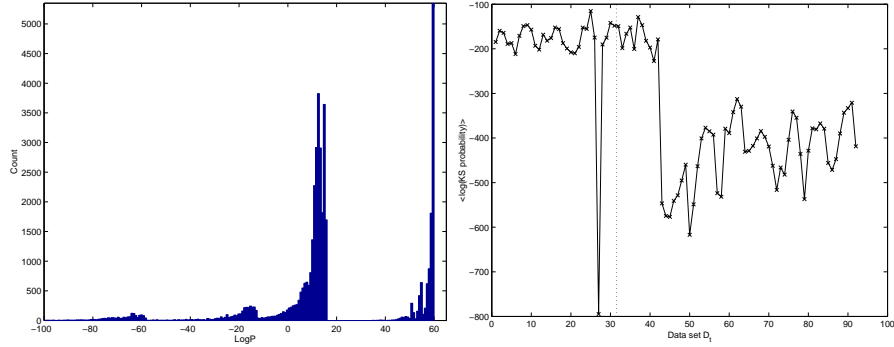

Figure 3: **Left:** distribution of 42655 $LogP$ scores from mixture of conditional indepen-
dence models. The data is a single-day of click-stream data from a commercial web site.
**Right:** Average log(KS probability) over the 31 day in-sample period for a content web-
site showing a *glitch* on day 27 and a permanent change on day 43, both detected by the
proposed methodology.

data sets (one month of data) that were used to build the initial model $\Theta_0$ can be used to
define the natural variability of the KS measures against which additional data sets can be
compared. The result is that in Figure 3 we clearly see a problem with the distribution on
day 27 (a *glitch* in the data) and a permanent change in the distribution on day 43. Both
of the detected changes correspond to real changes in the data, as verified by the commer-
cial website operators. Automatic description of changes in the distribution and criteria for
automatic rebuilding of the model are beyond scope of this paper.

## 4   Related Work

Automatic detection of various types of data changes appear in the literature in several
different flavors. For example, *novelty detection* ([4], [8]) is the task of determining unusual
or novel data points relative to some model. This is closely related to the *outlier detection*
problem ([1], [5]) where the goal is not only to find unusual data points, but the ones that
appear not to have been generated by the data generating distribution. A related problem
has been addressed by [2] in the context of time series modeling where outliers and trends
can contaminate the model estimation. More recently mixture models have been applied
more directly to outlier detection [3].

The method proposed in this paper addesses a different problem. We are not interested in
new and unusual data points; on the contrary, the method is quite robust with respect to
outliers. An outlier or two do not necessarily mean that the underlying data distribution has
changed. Also, some of the distribution changes we are interested in detecting might be
considered *uninteresting* and/or *not-novel*; for example, a slight shift of the population as
a whole is something that we certainly detect as a change but it is rarely considered novel
unless the shift is drastic.

There is also a set of online learning algorithms that update model parameters as the new
data becomes available (for variants and additional references, e.g. [6]). In that frame-
work there is no such concept as a data distribution change since the models are constantly
updated to reflect the most current distribution. For example, instead of detecting a slight
shift of the population as a whole, online learning algorithms update the model to reflect
the shift.

# 5 Conclusions

In this paper we introduced a model-based method for automatic distribution change detection in an online data environment. Given the $LogP$ distribution *data signature* we further showed how to compare different data sets relative to the model using KS statistics and how to obtain a single measure of similarity between the new data and the model. Finally, we discussed heuristics for change detection that become principled in the limit as the number of possible data sets increases.

Experimental results over synthetic and real online data indicate that the proposed methodology is able to alert the analyst to slight distributional changes. This methodology may be used as the basis of a system to automatically re-estimate parameters of a mixture model on an "as-needed" basis – when the model fails to adequately represent the data after a certain point in time.

## Footnotes

*Work was done while author was at digiMine, Inc., Bellevue, WA.

[1]This approach is called *maximum-likelihood* estimation. If we included parameter priors we could equally well apply results in this paper to the *maximum a posteriori* estimation.

# References

[1] V. Barnett and T. Lewis. *Outliers in statistical data*. Wiley, 1984.

[2] A. G. Bruce, J. T. Conor, and R. D. Martin. Prediction with robustness towards outliers, trends, and level shifts. In *Proceedings of the Third International Conference on Neural Networks in Financial Engineering*, pages 564–577, 1996.

[3] I. V. Cadez, P. Smyth, and H. Mannila. Probabilistic modeling of transaction data with applications to profiling, visualization, and prediction. In F. Provost and R. Srikant, editors, *Proceedings of the Seventh ACM SIGKDD International Conference on Knowledge Discovery and Data Mining*, pages 37–46. ACM, 2001.

[4] C. Campbell and K. P. Bennett. A linear programming approach to novelty detection. In T. K. Leen, T. G. Dietterich, and V. Tresp, editors, *Advances in Neural Information Processing Systems 13*, pages 395–401. MIT Press, 2001.

[5] T. Fawcett and F. J. Provost. Activity monitoring: Noticing interesting changes in behavior. In *Proceedings of the Fifth ACM SIGKDD International Conference on Knowledge Discovery and Data Mining*, pages 53–62, 1999.

[6] R. Neal and G. Hinton. A view of the em algorithm that justifies incremental, sparse and other variants. In M. I. Jordan, editor, *Learning in Graphical Models*, pages 355–368. Kluwer Academic Publishers, 1998.

[7] W. H. Press, S. A. Teukolsky, W. T. Vetterling, and B. P. Flannery. *Numerical Recipes in C: The Art of Scientific Computing, Second Edition*. Cambridge University Press, Cambridge, UK, 1992.

[8] B. Schölkopf, R. C. Williamson, A. J. Smola, J. Shawe-Taylor, and J. C. Platt. Support vector method for novelty detection. In S. A. Solla, T. K. Leen, and K.-R. Mller, editors, *Advances in Neural Information Processing Systems 12*, pages 582–588. MIT Press, 2000.
